# A Learning Analog Neural Network Chip with Continuous-Time Recurrent Dynamics

**Gert Cauwenberghs***
California Institute of Technology
Department of Electrical Engineering
128-95 Caltech, Pasadena, CA 91125
E-mail: gert@cco.caltech.edu

## Abstract

We present experimental results on supervised learning of dynamical features in an analog VLSI neural network chip. The recurrent network, containing six continuous-time analog neurons and 42 free parameters (connection strengths and thresholds), is trained to generate time-varying outputs approximating given periodic signals presented to the network. The chip implements a stochastic perturbative algorithm, which observes the error gradient along random directions in the parameter space for error-descent learning. In addition to the integrated learning functions and the generation of pseudo-random perturbations, the chip provides for teacher forcing and long-term storage of the volatile parameters. The network learns a 1 kHz circular trajectory in 100 sec. The chip occupies 2mm × 2mm in a 2μm CMOS process, and dissipates 1.2 mW.

## 1 Introduction

Exact gradient-descent algorithms for supervised learning in dynamic recurrent networks [1-3] are fairly complex and do not provide for a scalable implementation in a standard 2-D VLSI process. We have implemented a fairly simple and scalable

learning architecture in an analog VLSI recurrent network, based on a stochastic perturbative algorithm which avoids calculation of the gradient based on an explicit model of the network, but instead probes the dependence of the network error on the parameters directly [4]. As a demonstration of principle, we have trained a small network, integrated with the learning circuitry on a CMOS chip, to generate outputs following a prescribed periodic trajectory. The chip can be extended, with minor modifications to the internal structure of the cells, to accommodate applications with larger size recurrent networks.

## 2    System Architecture

The network contains six fully interconnected recurrent neurons with continuous-time dynamics,

$$\tau \frac{\mathrm{d}}{\mathrm{d}t} x_i = -x_i + \sum_{j=1}^{6} W_{ij} \, \sigma(x_j - \theta_j) + y_i \; , \tag{1}$$

with $x_i(t)$ the neuron states representing the outputs of the network, $y_i(t)$ the external inputs to the network, and $\sigma(.)$ a sigmoidal activation function. The 36 connection strengths $W_{ij}$ and 6 thresholds $\theta_j$ constitute the free parameters to be learned, and the time constant $\tau$ is kept fixed and identical for all neurons. Below, the parameters $W_{ij}$ and $\theta_j$ are denoted as components of a single vector $\mathbf{p}$.

The network is trained with target output signals $x_1^T(t)$ and $x_2^T(t)$ for the first two neuron outputs. Learning consists of minimizing the time-averaged error

$$\mathcal{E}(\mathbf{p}) = \lim_{T \to \infty} \frac{1}{2T} \int_{-T}^{T} \sum_{k=1}^{2} |x_k^T(t) - x_k(t)|^{\nu} \mathrm{d}t \; , \tag{2}$$

using a distance metric with norm $\nu$. The learning algorithm [4] iteratively specifies incremental updates in the parameter vector $\mathbf{p}$ as

$$\mathbf{p}^{(k+1)} = \mathbf{p}^{(k)} - \mu \, \hat{\mathcal{E}}^{(k)} \, \boldsymbol{\pi}^{(k)} \tag{3}$$

with the perturbed error

$$\hat{\mathcal{E}}^{(k)} = \frac{1}{2} \left( \mathcal{E}(\mathbf{p}^{(k)} + \boldsymbol{\pi}^{(k)}) - \mathcal{E}(\mathbf{p}^{(k)} - \boldsymbol{\pi}^{(k)}) \right) \tag{4}$$

obtained from a two-sided parallel activation of fixed-amplitude random perturbations $\pi_i^{(k)}$ onto the parameters $p_i^{(k)}$; $\pi_i^{(k)} = \pm\sigma$ with equal probabilities for both polarities. The algorithm basically performs random-direction descent of the error as a multi-dimensional extension to the Kiefer-Wolfowitz stochastic approximation method [5], and several related variants have recently been proposed for optimization [6,7] and hardware learning [8-10].

To facilitate learning, a teacher forcing signal is initially applied to the external input y according to

$$y_i(t) = \lambda \, \gamma(x_i^T(t) - x_i(t)) \; , \quad i = 1, 2 \tag{5}$$

providing a feedback mechanism that forces the network outputs towards the targets [3]. A symmetrical and monotonically increasing "squashing" function for $\gamma(.)$ serves this purpose. The teacher forcing amplitude $\lambda$ needs to be attenuated along the learning process, as to suppress the bias in the network outputs at convergence that might result from residual errors.

# 3   Analog VLSI Implementation

The network and learning circuitry are implemented on a single analog CMOS chip, which uses a transconductance current-mode approach for continuous-time operation. Through dedicated transconductance circuitry, a wide linear dynamic range for the voltages is achieved at relatively low levels of power dissipation (experimentally 1.2 mW while either learning or refreshing). While most learning functions, including generation of the pseudo-random perturbations, are integrated on-chip in conjunction with the network, some global and higher-level learning functions of low dimensionality, such as the evaluation of the error (2) and construction of the perturbed error (4), are performed outside the chip for greater flexibility in tailoring the learning process. The structure and functionality of the implemented circuitry are illustrated in Figures 1 to 3, and a more detailed description follows below.

## 3.1   Network Circuitry

Figure 1 shows the schematics of the synapse and neuron circuitry. A synapse cell of single polarity is shown in Figure 1 (a). A high output impedance triode multiplier, using an adjustable regulated cascode [11], provides a constant current $I_{ij}$ linear in the voltage $W_{ij}$ over a wide range. The synaptic current $I_{ij}$ feeds into a differential pair, injecting a differential current $I_{ij} \ \sigma(x_j - \theta_j)$ into the diode-connected $I_{out}^+$ and $I_{out}^-$ output lines. The double-stack transistor configuration of the differential pair offers an expanded linear sigmoid range. The summed output currents $I_{out}^+$ and $I_{out}^-$ of a row of synapses are collected in the output cell, Figure 1 (b), which also subtracts the reference currents $I_{ref}^+$ and $I_{ref}^-$ obtained from a reference row of "dummy" synapses defining the "zero-point" synaptic strength $W_{off}$ for bipolar operation. The thus established current corresponds to the summed synaptic contributions in (1). Wherever appropriate $(i = 1, 2)$, a differential transconductance element with inputs $x_i$ and $x_i^T$ is added to supply an external input current for forced teacher action in accordance with (5).

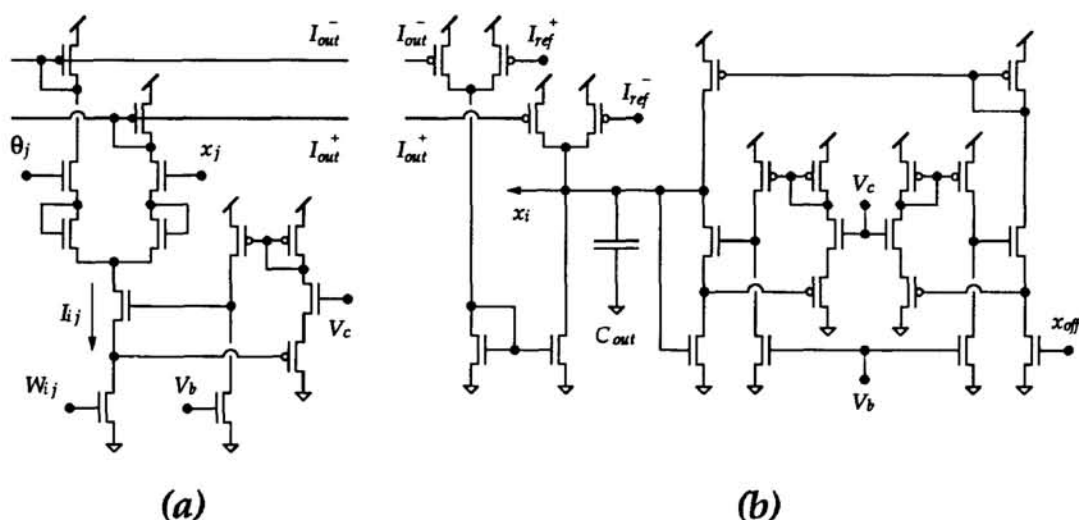

*(a)*                                                              *(b)*

**Figure 1** Schematics of synapse and neuron circuitry. *(a)* Synapse of single polarity. *(b)* Output cell with current-to-voltage converter.

The output current is converted to the neuron output voltage $x_i$, through an active resistive element using the same regulated high output impedance triode circuitry as used in the synaptic current source. The feedback delay parameter $\tau$ in (1) corresponds to the $RC$

product of the regulated triode active resistance value and the capacitance $C_{out}$. With $C_{out} = 5$ pF, the delay ranges between 20 and $200 \mu sec$, adjustable by the control voltage of the regulated cascode. Figure 2 shows the measured static characteristics of the synapse and neuron functions for different values of $W_{ij}$ and $\theta_j$ ( $i = j = 1$), obtained by disabling the neuron feedback and driving the synapse inputs externally.

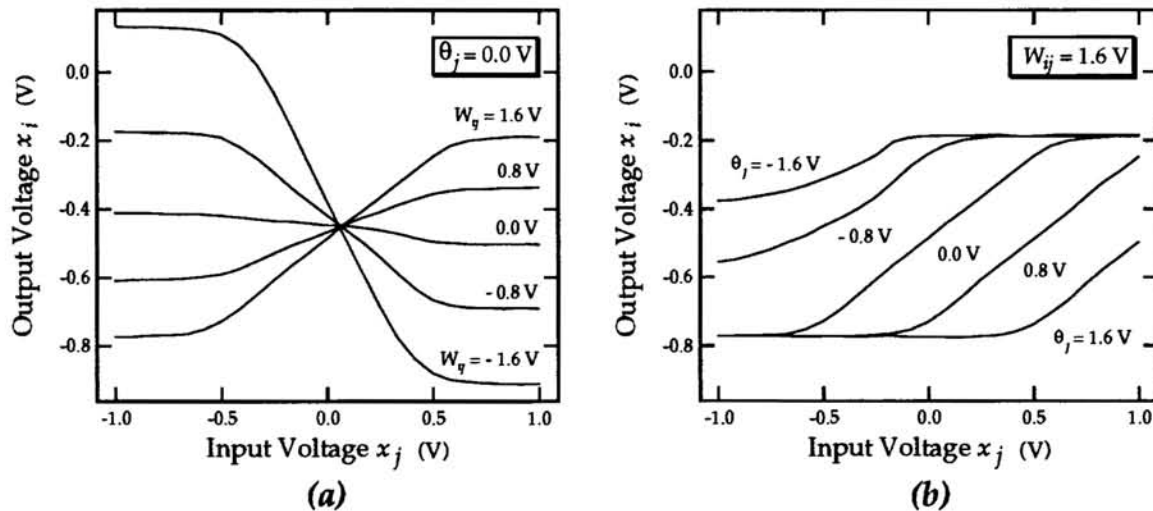

**Figure 2** Measured static synapse and neuron characteristics, for various values of
*(a)* the connection strength $W_{ij}$, and   *(b)* the threshold $\theta_j$.

## 3.2   Learning Circuitry

Figure 3 (a) shows the simplified schematics of the learning and storage circuitry, replicated locally for every parameter (connection strength or threshold) in the network. Most of the variables relating to the operation of the cells are local, with exception of a few global signals communicating to all cells. Global signals include the sign and the amplitude of the perturbed error $\hat{\mathcal{E}}$ and predefined control signals. The stored parameter and its binary perturbation are strictly local to the cell, in that they do not need to communicate explicitly to outside circuitry (except trivially through the neural network it drives), which simplifies the structural organization and interconnection of the learning cells.

The parameter voltage $p_i$ is stored on the capacitor $C_{store}$, which furthermore couples to capacitor $C_{pert}$ for activation of the perturbation. The perturbation bit $\pi_i$ selects either of two complementary signals $V_{+\sigma}$ and $V_{-\sigma}$ with corresponding polarity. With the specific shape of the waveforms $V_{+\sigma}$ and $V_{-\sigma}$ depicted in Figure 3 (b), the proper sequence of perturbation activations is established for observation of the complementary error terms in (4). The obtained global value for $\hat{\mathcal{E}}$ is then used, in conjunction with the local perturbation bit $\pi_i$, to update the parameter value $p_i$ according to (3). A fine-resolution charge-pump, shown in the dashed-line inset of Figure 3 (a), is used for this purpose. The charge pump dumps either of a positive or negative update current, of equal amplitude, onto the storage capacitor whenever it is activated by means of an EN_UPD high pulse, effecting either of a given increment or decrement on the parameter value $p_i$ respectively. The update currents are supplied by two complementary transistors, and are switched by driving the source voltages of the transistors rather than their gate voltages in order to avoid typical clock feed-through effects. The amplitude of the incremental update, set proportionally to $|\hat{\mathcal{E}}|$, is controlled by the $V_{UPD\ n}$ and $V_{UPD\ p}$ gate voltage levels, operated in the sub-threshold region. The polarity of the increment or decrement action is determined by the control signal DECR/$\overline{INCR}$, obtained from the polarities of

the perturbed error $\hat{\mathcal{E}}$ and the perturbation bit $\pi_i$ through an exclusive-or operation. The learning cycle is completed by activating the update by a high pulse on EN_UPD. The next learning cycle then starts with a new random bit value for the perturbation $\pi_i$.

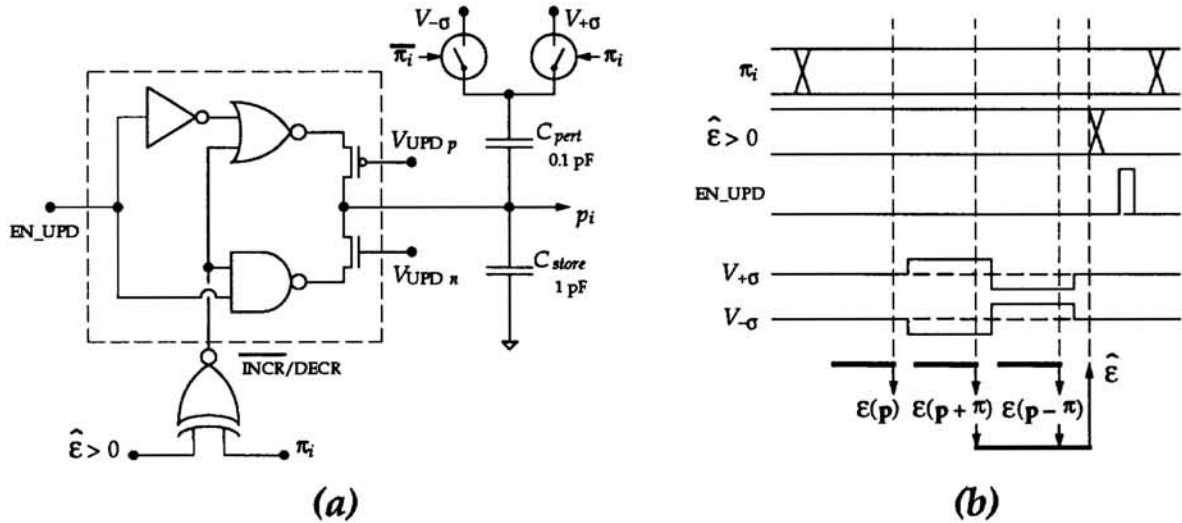

**(a)**                                    **(b)**

**Figure 3** Learning cell circuitry. *(a)* Simplified schematics.
*(b)* Waveform and timing diagram.

The random bit stream $\pi_i^{(k)}$ is generated on-chip by means of a set of linear feedback shift registers [12]. For optimal performance, the perturbations need to satisfy certain statistical orthogonality conditions, and a rigorous but elaborate method to generate a set of uncorrelated bit streams in VLSI has been derived [13]. To preserve the scalability of the learning architecture and the local nature of the perturbations, we have chosen a simplified scheme which does not affect the learning performance to first order, as verified experimentally. The array of perturbation bits, configured in a two-dimensional arrangement as prompted by the location of the parameters in the network, is constructed by an outer-product exclusive-or operation from two generating linear sets of uncorrelated row and column bits on lines running horizontally and vertically across the network array.

In the present implementation the evaluation of the error functional (2) is performed externally with discrete analog components, leaving some flexibility to experiment with different formulations of error functionals that otherwise would have been hardwired. A mean absolute difference ($\nu = 1$) norm is used for the metric distance, and the time-averaging of the error is achieved by a fourth-order Butterworth low-pass filter. The cut-off frequency is tuned to accommodate an AC ripple smaller than 0.1%, giving rise to a filter settling time extending 20 periods of the training signal.

## 3.3   Long-Term Volatile Storage

After learning, it is desirable to retain ("freeze") the learned information, in principle for an infinite period of time. The volatile storage of the parameter values on capacitors undergoes a spontaneous decay due to junction leakage and other drift phenomena, and needs to be refreshed periodically. For eight effective bits of resolution, a refresh rate of 10 Hz is sufficient. Incidentally, the charge pump used for the learning updates provides for refresh of the parameter values as well. To that purpose, probing and multiplexing circuitry (not shown) are added to the learning cell of Figure 3 (a) for sequential refresh. In the experiment conducted here, the parameters are stored externally and refreshed sequentially by activating the corresponding charge pump with a DECR/$\overline{\text{INCR}}$ bit defined by the polarity of the observed deviation between internally probed and externally stored

values. The parameter refresh is performed in the background with a 100 msec cycle, and does not interfere with the continuous-time network operation. A simple internal analog storage method obliterating the need of external storage is described in [14], and is supported by the chip architecture.

# 4  Learning Experiment

As a proof of principle, the network is trained with a circular target trajectory defined by the quadrature-phase oscillator

$$\begin{cases} x_1^T(t) &= A\cos{(2\pi ft)} \\ x_2^T(t) &= A\sin{(2\pi ft)} \end{cases} \tag{6}$$

with $A = 0.8\text{V}$ and $f = 1\text{kHz}$. In principle a recurrent network of two neurons suffices to generate quadrature-phase oscillations, and the extra neurons in the network serve to accommodate the particular amplitude and frequency requirements and assist in reducing the nonlinear harmonic distortion.

Clearly the initial conditions for the parameter values distinguish a trivial learning problem from a hard one, and training an arbitrarily initialized network may lead to unpredictable results of poor generality. Incidentally, we found that the majority of randomly initialized learning sessions fail to generate oscillatory behavior at convergence, the network being trapped in a local minimum defined by a strong point attractor. Even with strong teacher forcing these local minima persist. In contrast, we obtained consistent and satisfactory results with the following initialization of network parameters: strong positive diagonal connection strengths $W_{ii} = 1$, zero off-diagonal terms $W_{ij} = 0$ ; $i \neq j$ and zero thresholds $\theta_i = 0$. The positive diagonal connections $W_{ii}$ repel the neuron outputs from the point attractor at the origin, counteracting the spontaneous decay term $-x_i$ in (1). Applying non-zero initial values for the cross connections $W_{ij}$ ; $i \neq j$ would introduce a bias in the dynamics due to coupling between neurons. With zero initial cross coupling, and under strong initial teacher forcing, fairly fast and robust learning is achieved.

Figure 4 shows recorded error sequences under training of the network with the target oscillator (6), for five different sessions of $1,500$ learning iterations each starting from the above initial conditions. The learning iterations span 60 msec each, for a total of 100 sec per session. The teacher forcing amplitude $\lambda$ is set initially to 3 V, and thereafter decays logarithmically over one order of magnitude towards the end of the sessions. Fixed values of the learning rate and the perturbation amplitude are used throughout the sessions, with $\mu = 25.6 \text{ V}^{-1}$ and $\sigma = 12.5 \text{ mV}$. All five sessions show a rapid initial decrease in the error under stimulus of the strong teacher forcing, and thereafter undergo a region of persistent flat error slowly tapering off towards convergence as the teacher forcing is gradually released. Notice that this flat region does not imply slow learning; instead the learning constantly removes error as additional error is adiabatically injected by the relaxation of the teacher forcing.

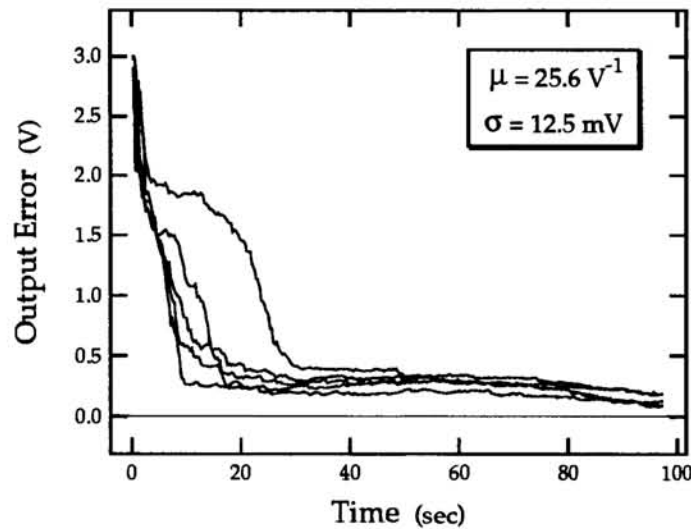

**Figure 4** Recorded evolution of the error during learning,
for five different sessions on the network.

Near convergence, the bias in the network error due to the residual teacher forcing becomes negligible. Figure 5 shows the network outputs and target signals at convergence, with the learning halted and the parameter refresh activated, illustrating the minor effect of the residual teacher forcing signal on the network dynamics. The oscillogram of Figure 5 (a) is obtained under a weak teacher forcing signal, and that of Figure 5 (b) is obtained with the same network parameters but with the teacher forcing signal disabled. In both cases the oscilloscope is triggered on the network output signals. Obviously, in absence of teacher forcing the network does no longer run synchronously with the target signal. However, the discrepancy in frequency, amplitude and shape between either of the free-running and forced oscillatory output waveforms and the target signal waveforms is evidently small.

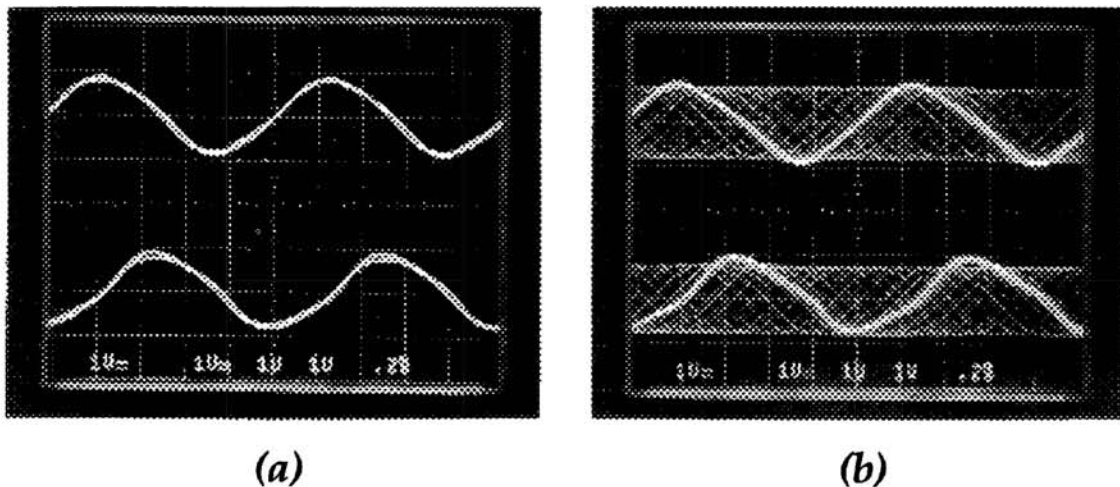

$(a)$                                $(b)$

**Figure 5** Oscillograms of the network outputs and target signals after learning,
$(a)$ under weak teacher forcing, and $(b)$ with teacher forcing disabled.
Top traces: $x_1(t)$ and $x_1^T(t)$. Bottom traces: $x_2(t)$ and $x_2^T(t)$.

# 5    Conclusion

We implemented a small-size learning recurrent neural network in an analog VLSI chip, and verified its learning performance in a continuous-time setting with a simple dynamic test (learning of a quadrature-phase oscillator). By virtue of its scalable architecture, with constant requirements on interconnectivity and limited global communication, the network structure with embedded learning functions can be freely expanded in a two-dimensional arrangement to accommodate applications of recurrent dynamical networks requiring larger dimensionality. A present limitation of the implemented learning model is the requirement of periodicity on the input and target signals during the learning process, which is needed to allow a repetitive and consistent evaluation of the network error for the parameter updates.

**Acknowledgments**

Fabrication of the CMOS chip was provided through the DARPA/NSF MOSIS service. Financial support by the NIPS Foundation largely covered the expenses of attending the conference.

**References**

[1]  B.A. Pearlmutter, "Learning State Space Trajectories in Recurrent Neural Networks," *Neural Computation*, vol. **1** (2), pp 263-269, 1989.

[2]  R.J. Williams and D. Zipser, "A Learning Algorithm for Continually Running Fully Recurrent Neural Networks," *Neural Computation*, vol. **1** (2), pp 270-280, 1989.

[3]  N.B. Toomarian, and J. Barhen, "Learning a Trajectory using Adjoint Functions and Teacher Forcing," *Neural Networks*, vol. **5** (3), pp 473-484, 1992.

[4]  G. Cauwenberghs, "A Fast Stochastic Error-Descent Algorithm for Supervised Learning and Optimization," in *Advances in Neural Information Processing Systems*, San Mateo, CA: Morgan Kaufman, vol. **5**, pp 244-251, 1993.

[5]  H.J. Kushner, and D.S. Clark, "Stochastic Approximation Methods for Constrained and Unconstrained Systems," New York, NY: Springer-Verlag, 1978.

[6]  M.A. Styblinski, and T.-S. Tang, "Experiments in Nonconvex Optimization: Stochastic Approximation with Function Smoothing and Simulated Annealing," *Neural Networks*, vol. **3** (4), pp 467-483, 1990.

[7]  J.C. Spall, "Multivariate Stochastic Approximation Using a Simultaneous Perturbation Gradient Approximation," *IEEE Trans. Automatic Control*, vol. **37** (3), pp 332-341, 1992.

[8]  J. Alspector, R. Meir, B. Yuhas, and A. Jayakumar, "A Parallel Gradient Descent Method for Learning in Analog VLSI Neural Networks," in *Advances in Neural Information Processing Systems*, San Mateo, CA: Morgan Kaufman, vol. **5**, pp 836-844, 1993.

[9]  B. Flower and M. Jabri, "Summed Weight Neuron Perturbation: An $\mathcal{O}(n)$ Improvement over Weight Perturbation," in *Advances in Neural Information Processing Systems*, San Mateo, CA: Morgan Kaufman, vol. **5**, pp 212-219, 1993.

[10]  D. Kirk, D. Kerns, K. Fleischer, and A. Barr, "Analog VLSI Implementation of Gradient Descent," in *Advances in Neural Information Processing Systems*, San Mateo, CA: Morgan Kaufman, vol. **5**, pp 789-796, 1993.

[11]  J.W. Fattaruso, S. Kiriaki, G. Warwar, and M. de Wit, "Self-Calibration Techniques for a Second-Order Multibit Sigma-Delta Modulator," in *ISSCC Technical Digest*, IEEE Press, vol. **36**, pp 228-229, 1993.

[12]  S.W. Golomb, "Shift Register Sequences," San Francisco, CA: Holden-Day, 1967.

[13]  J. Alspector, J.W. Gannett, S. Haber, M.B. Parker, and R. Chu, "A VLSI-Efficient Technique for Generating Multiple Uncorrelated Noise Sources and Its Application to Stochastic Neural Networks," *IEEE T. Circuits and Systems*, **38** (1), pp 109-123, 1991.

[14]  G. Cauwenberghs, and A. Yariv, "Method and Apparatus for Long-Term Multi-Valued Storage in Dynamic Analog Memory," U.S. Patent pending, filed 1993.

## Footnotes

*Present address: Johns Hopkins University, ECE Dept., Baltimore MD 21218-2686.
